# Risk Sensitive Reinforcement Learning

**Ralph Neuneier**
Siemens AG, Corporate Technology
D-81730 München, Germany
Ralph.Neuneier@mchp.siemens.de

**Oliver Mihatsch**
Siemens AG, Corporate Technology
D-81730 München, Germany
Oliver.Mihatsch@mchp.siemens.de

## Abstract

As already known, the expected return of a policy in Markov Decision Problems is not always the most suitable optimality criterion. For many applications control strategies have to meet various constraints like avoiding very bad states (risk-avoiding) or generating high profit within a short time (risk-seeking) although this might probably cause significant costs. We propose a modified $Q$-learning algorithm which uses a single continuous parameter $\kappa \in [-1, 1]$ to determine in which sense the resulting policy is optimal. For $\kappa = 0$, the policy is optimal with respect to the usual expected return criterion, while $\kappa \to 1$ generates a solution which is optimal in worst case. Analogous, the closer $\kappa$ is to $-1$ the more risk seeking the policy becomes. In contrast to other related approaches in the field of MDPs we do not have to transform the cost model or to increase the state space in order to take risk into account. Our new approach is evaluated by computing optimal investment strategies for an artificial stock market.

## 1 WHY IT SOMETIMES PAYS TO ACT CAUTIOUSLY

Reinforcement learning (RL) deals with the computation of favorable control policies in sequential decision task. Its theoretical framework of Markov Decision Problems (MDPs) evaluates and compares policies by their expected (sometimes discounted or averaged) sum of the immediate returns or costs per time step (Bertsekas & Tsitsiklis, 1996). But there are numerous applications which require a more sophisticated control scheme: e. g. a policy should take into account that bad outcomes or states may be possible even if they are very rare because they are so disastrous, that they should be certainly avoided.

An obvious example is the field of finance where the main question is how to invest resources among various opportunities (e.g. assets like stocks, bonds, etc.) to achieve remarkable returns while simultaneously controlling the risk exposure of the investments due to changing markets or economic conditions. Many traders try to achieve this by a Markovitz-like portfolio management which distributes capital according to return and risk

estimates of the assets. A new approach using reinforcement learning techniques which additionally integrates trading costs and other market imperfections has been proposed in Neuneier, 1998. Here, these algorithms are naturally extended such that an explicit risk control is now possible. The investor can decide how much risk she/he is willing to accept and then compute an optimal risk-averse investment strategy. Similar trade-off scenarios can be formulated in robotics, traffic control and further application areas.

The fact that the popular expected value criterion is not always suitable has been already known in the field of AI (Koenig & Simmons, 1994), control theory and reinforcement learning (Heger, 1994 and Szepesvári, 1997). Several techniques have been proposed to handle this problem. The most obvious way is to transform the sum of returns $\sum_t r_t$ using an appropriate utility function $U$ which reflects the desired properties of the solution. Unfortunately, interesting nonlinear utility functions incorporating the variance of the return, such as $U(\sum_t r_t) = \sum_t r_t - \lambda(\sum_t r_t - E(\sum_t r_t))^2$, lead to non-Markovian decision problems. The popular class of exponential utility functions $U(\sum_t r_t) = \exp(\lambda \sum_t r_t)$ preserves the Markov property but requires time dependent policies even for discounted infinite horizon MDPs. Furthermore, it is not possible to formulate a corresponding model-free learning algorithm. A further alternative changes the state space model by including past returns as an additional state element at the cost of a higher dimensionality of the MDP. Furthermore, it is not always clear in which way the states should be augmented. One may also transform the cost model, i.e. by punishing large losses stronger than minor costs. While requiring a significant amount of prior knowledge, this also increases the complexity of the MDP.

In contrast to these approaches we modify the popular $Q$-learning algorithm by introducing a control parameter which determines in which sense the resulting policy is optimal. Intuitively and loosely speaking, our algorithm simulates the learning behavior of an optimistic (pessimistic) person by overweighting (underweighting) experiences which are more positive (negative) than expected. This main idea will be made more precise in section 2 and mathematically thoroughly analyzed in section 3. Using artificial data, we demonstrate some properties of the new algorithm by constructing an optimal risk-avoiding investment strategy (section 4).

## 2  RISK SENSITIVE Q-LEARNING

For brevity we restrict ourselves to the subclass of infinite horizon discounted Markov decision problems (MDP). Furthermore, we assume the immediate rewards being deterministic functions of the current state and control action. Let $S = \{1, \ldots, n\}$ be the finite state space and $U$ be the finite action space. Transition probabilities and immediate rewards are denoted by $p_{ij}(u)$ and $g_i(u)$, respectively. $\gamma$ denotes the discount factor. Let $\Pi$ be the set of all deterministic policies mapping states to control actions.

A commonly used objective is to learn a policy $\pi$ that

$$\text{maximizes } \left( \overline{Q}^\pi(i, u) := g_i(u) + E \left\{ \sum_{k=1}^\infty \gamma^k g_{i_k}(\pi(i_k)) \right\} \right) \tag{1}$$

quantifying the expected reward if one executes control action $u$ in state $i$ and follows the policy $\pi$ thereafter. It is a well-known result that the optimal $Q$-values $\overline{Q}^*(i, u) := max_{\pi \in \Pi} \overline{Q}^\pi(i, u)$ satisfy the following optimality equation

$$\overline{Q}^*(i, u) = g_i(u) + \gamma \sum_{j \in S} p_{ij}(u) \max_{u' \in U} \overline{Q}^*(j, u') \quad \forall i \in S, u \in U. \tag{2}$$

Any policy $\overline{\pi}$ with $\overline{\pi}(i) = \arg\max_{u \in U} \overline{Q}^*(i, u)$ is optimal with respect to the expected reward criterion.

The $Q$-function $\overline{Q}^{\pi}$ averages over the outcome of all possible trajectories (series of states) of the Markov process generated by following the policy $\pi$. However, the outcome of a specific realization of the Markov process may deviate significantly from this mean value. The expected reward criterion does not consider any risk, although the cases where the discounted reward falls considerably below the mean value is of a living interest for many applications. Therefore, depending on the application at hand the expected reward approach is not always appropriate. Alternatively, Heger (1994) and Littman & Szepesvári (1996) present a performance criterion that exclusively focuses on risk avoiding policies:

$$\text{maximize} \left( \underline{Q}^{\pi}(i,u) := g_i(u) + \inf_{\substack{i_1,i_2,\dots \\ p(i_1,i_2,\dots)>0}} \left\{ \sum_{k=1}^{\infty} \gamma^k g_{i_k}(\pi(i_k)) \right\} \right). \tag{3}$$

The $Q$-function $\underline{Q}^{\pi}(i,u)$ denotes the worst possible outcome if one executes control action $u$ in state $i$ and follows the policy $\pi$ thereafter. The corresponding optimality equation for $\underline{Q}^{*}(i,u) := \max_{\pi \in \Pi} \underline{Q}^{\pi}(i,u)$ is given by

$$\underline{Q}^{*}(i,u) = g_i(u) + \gamma \min_{\substack{j \in S \\ p_{ij}(u)>0}} \max_{u' \in U} \underline{Q}^{*}(j,u'). \tag{4}$$

Any policy $\underline{\pi}$ satisfying $\underline{\pi}(i) = \arg\max_{u \in U} \underline{Q}^{*}(i,u)$ is optimal with respect to this minimal reward criterion. In most real world applications this approach is too restrictive because it takes very rare events (that in practice never happen) fully into account. This usually leads to policies with a lower average performance than the application requires. An investment manager, for instance, which acts with respect to this very pessimistic objective function will not invest at all.

To handle the trade-off between a sufficient average performance and a risk avoiding (risk seeking) behavior, we propose a family of new optimality equations parameterized by a meta-parameter $\kappa$ $(-1 < \kappa < 1)$:

$$0 = \sum_{j \in S} p_{ij}(u) \mathcal{X}^{\kappa} \left( g_i(u) + \gamma \max_{u' \in U} Q_{\kappa}(j,u') - Q_{\kappa}(i,u) \right) \quad \forall i \in S, u \in U \tag{5}$$

where $\mathcal{X}^{\kappa}(x) := (1 - \kappa \, \text{sign}(x))x$. (In the next section we will show that a unique solution $Q_{\kappa}$ of the above equation (5) exists.) Obviously, for $\kappa = 0$ we recover equation (2), the optimality equation for the expected reward criterion. If we choose $\kappa$ to be positive $(0 < \kappa < 1)$ then we overweight negative temporal differences

$$g_i(u) + \gamma \max_{u' \in U} Q_{\kappa}(j,u') - Q_{\kappa}(i,u) < 0 \tag{6}$$

with respect to positive ones. Loosely speaking, we overweight transitions to states where the future return is lower than the average one. On the other hand, we underweight transitions to states that promise a higher return than in the average. Thus, an agent that behaves according to the policy $\pi_{\kappa}(i) := \arg\max_{u \in U} Q_{\kappa}(i,u)$ is risk avoiding if $\kappa > 0$. In the limit $\kappa \to 1$ the policy $\pi_{\kappa}$ approaches the optimal worst-case policy $\underline{\pi}$, as we will show in the following section. (To get an intuition about this, the reader may easily check that the optimal worst-case $Q$-value $\underline{Q}^{*}$ fulfills the modified optimality equation (5) for $\kappa = 1$.) Similarly, the policy $\pi_{\kappa}$ becomes risk seeking if we choose $\kappa$ to be negative.

It is straightforward to formulate a risk sensitive $Q$-learning algorithm that bases on the modified optimality equation (5). Let $\hat{Q}_{\kappa}(i,u;w)$ be a parametric approximation of the Q-function $Q_{\kappa}(i,u)$. The states and actions encountered at time step $k$ during simulation are denoted by $i_k$ and $u_k$ respectively. At each time step apply the following update rule:

$$d^{(k)} = g_{i_k}(u_k) + \gamma \max_{u' \in U} \hat{Q}_{\kappa}(i_{k+1},u';w^{(k)}) - \hat{Q}_{\kappa}(i_k,u_k;w^{(k)}),$$

$$w^{(k+1)} = w^{(k)} + \alpha_{\kappa}^{(k)} \mathcal{X}^{\kappa}(d^{(k)}) \nabla_w \hat{Q}_{\kappa}(i_k,u_k;w^{(k)}), \tag{7}$$

where $\alpha_\kappa^{(k)}$ denotes a stepsize sequence. The following section analyzes the properties of the new optimality equations and the corresponding $Q$-learning algorithm.

## 3    PROPERTIES OF THE RISK SENSITIVE Q-FUNCTION

Due to space limitations we are not able to give detailed proofs of our results. Instead, we focus on interpreting their practical consequences. The proofs will be published elsewhere.

Before formulating the mathematical results, we introduce some notation to make the exposition more concise. Using an arbitrary stepsize $0 < \alpha < 1$, we define the value iteration operator corresponding to our modified optimality equation (5) as

$$\mathcal{T}_{\alpha,\kappa}[Q](i,u) := Q(i,u) + \alpha \sum_{j \in S} p_{ij}(u)\mathcal{X}^\kappa\Big(g_i(u) + \gamma \max_{u' \in U} Q(j,u') - Q(i,u)\Big). \quad (8)$$

The operator $\mathcal{T}_{\alpha,\kappa}$ acts on the space of $Q$-functions. For every $Q$-function $Q$ and every state-action pair $(i,u)$ we define $N_\kappa[Q](i,u)$ to be the set of all successor states $j$ for which $\max_{u' \in U} Q(j,u')$ attains its minimum:

$$N_\kappa[Q](i,u) := \Big\{ j \in S \,|\, p_{ij}(u) > 0 \text{ and } \max_{u' \in U} Q(j,u') = \min_{\substack{j' \in S \\ p_{ij'}(u) > 0}} \max_{u' \in U} Q(j',u') \Big\}. \quad (9)$$

Let $p_\kappa[Q](i,u) := \sum_{j \in N_\kappa[Q](i,u)} p_{ij}(u)$ be the probability of transitions to such successor states.

We have the following lemma ensuring the contraction property of $\mathcal{T}_{\alpha,\kappa}$.

**Lemma 1 (Contraction Property)** *Let* $|Q| = \max_{i \in S, u \in U} Q(i,u)$ *and* $0 < \alpha < 1$, $0 < \gamma < 1$. *Then*

$$|\mathcal{T}_{\alpha,\kappa}[Q_1] - \mathcal{T}_{\alpha,\kappa}[Q_2]| \le (1 - \alpha(1 - |\kappa|)(1 - \gamma)) |Q_1 - Q_2| \quad \forall Q_1, Q_2. \quad (10)$$

*The operator* $\mathcal{T}_{\alpha,\kappa}$ *is contracting, because* $0 < (1 - \alpha(1 - |\kappa|)(1 - \gamma)) < 1$.

The lemma has several important consequences.

1. The risk sensitive optimality equation (5), i. e. $\mathcal{T}_{\alpha,\kappa}[Q] = Q$ has a unique solution $Q_\kappa$ for all $-1 < \kappa < 1$.

2. The value iteration procedure $Q_{\text{new}} := \mathcal{T}_{\alpha,\kappa}[Q]$ converges towards $Q_\kappa$.

3. The existing convergence results for traditional $Q$-learning (Bertsekas & Tsitsiklis 1997, Tsitsiklis & Van Roy 1997) remain also valid in the risk sensitive case $\kappa \neq 0$. Particularly, risk sensitive $Q$-learning (7) converges with probability one in the case of lookup table representations as well as in the case of optimal stopping problems combined with linear representations.

4. The speed of convergence for both, risk sensitive value iteration and $Q$-learning becomes worse if $|\kappa| \to 1$. We can remedy this to some extent if we increase the stepsize $\alpha$ appropriately.

Let $\pi_\kappa$ be a greedy policy with respect to the unique solution $Q_\kappa$ of our modified optimality equation; that is $\pi_\kappa(i) = \arg\max_{u \in U} Q_\kappa(i,u)$. The following theorem examines the performance of $\pi_\kappa$ for the risk avoiding case $\kappa \ge 0$. It gives us a feeling about the expected outcome $\overline{Q}^{\pi_\kappa}$ and the worst possible outcome $\underline{Q}^{\pi_\kappa}$ of policy $\pi_\kappa$ for different values of $\kappa$. The theorem clarifies the limiting behavior of $\pi_\kappa$ if $\kappa \to 1$.

**Theorem 2** *Let* $0 \leq \kappa < 1$. *The following inequalities hold componentwise, i. e. for each pair* $(i, u) \in S \times U$.

$$0 \leq \overline{Q}^* - \overline{Q}^{\pi_\kappa} \leq 2\kappa \frac{\gamma}{1-\gamma}(\overline{Q}^* - \underline{Q}^*) \tag{11}$$

$$0 \leq p_\kappa[Q_\kappa](\underline{Q}^* - \underline{Q}^{\pi_\kappa}) \leq \frac{(1-\kappa)}{2\kappa} \frac{\gamma}{1-\gamma}(\overline{Q}^* - \underline{Q}^*) \tag{12}$$

*Moreover,* $\lim\limits_{\kappa \to 0} \overline{Q}^{\pi_\kappa} = \overline{Q}^*$ *and* $\lim\limits_{\kappa \to 1} \underline{Q}^{\pi_\kappa} = \underline{Q}^*$.

The difference $\overline{Q}^* - \underline{Q}^*$ between the optimal expected reward and the optimal worst case reward is crucial in the above inequalities. It measures the amount of risk being inherent in our MDP at hand. Besides the value of $\kappa$, this quantity essentially influences the difference between the performance of the policy $\pi_\kappa$ and the optimal performance with respect to both, the expected reward and the worst case criterion. The second inequality (12) states that the performance of policy $\pi_\kappa$ in the worst case sense tends to the optimal worst case performance if $\kappa \to 1$. The "speed of convergence" is influenced by the quantity $p_\kappa[Q_\kappa]$, i. e. the probability that a worst case transition really occurs. (Note that $p_\kappa[Q_\kappa]$ is bounded from below.) A higher probability $p_\kappa[Q_\kappa]$ of worst case transitions implies a stronger risk avoiding attitude of the policy $\pi_\kappa$.

## 4   EXPERIMENTS: RISK-AVERSE INVESTMENT DECISIONS

Our algorithm is now tested on the task of constructing an optimal investment policy for an artificial stock price analogous to the empirical analysis in Neuneier, 1998. The task, illustrated as a MDP in fig. 1, is to decide at each time step (e. g. each day or after each mayor event on the market) whether to buy the stock and therefore speculating on increasing stock prices or to keep the capital in cash which avoids potential losses due to decreasing stock prices.

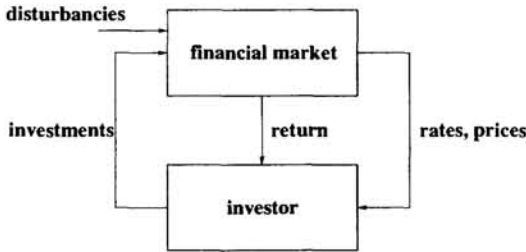

**Figure 1.  The Markov Decision Problem:**

$x_t = (\$_t, K_t)'$   state: market $\$_t$ and portfolio $K_t$
$a_t = \mu(x_t)$   policy $\mu$, actions
$p(x_{t+1}|x_t)$   transition probabilities
$r(x_t, a_t, \$_{t+1})$   return function

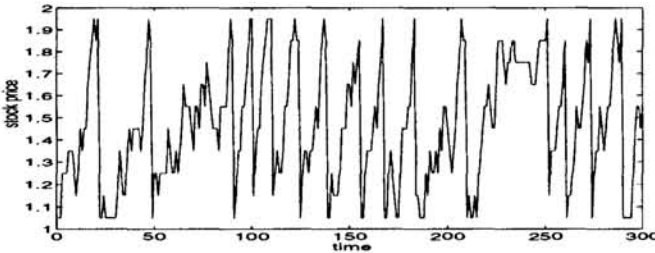

**Figure 2.** A realization of the artificial stock price for 300 time steps.  It is obvious that the price follows an increasing trend but with higher values a sudden drop to low values becomes more and more probable.

It is assumed, that the investor is not able to influence the market by the investment decisions.  This leads to a MDP with some of the state elements being uncontrollable and results in two computationally import implications: first, one can simulate the investments by historical data without investing (and potentially losing) real money. Second, one can formulate a very efficient (memory saving) and more robust $Q$-learning algorithms. Due to space restriction we skip a detailed description of these algorithms and refer the interested reader to Neuneier, 1998.

The artificial stock price is in the range of $[1, 2]$. The transition probabilities are chosen such that the stock market simulates a situation where the price follows an increasing trend but with higher values a drop to very low values becomes more and more probable (fig. 2).

The state vector consists of the current stock price and the current investment, i.e. the amount of money invested in stocks or cash. Changing the investment from cash to stocks results in some transaction costs consisting of variable and fixed terms. These costs are essential to define the investment problem as a MDP because they couple the actions made at different time steps. Otherwise we could solve the problem by a pure prediction of the next stock price. The function which quantifies the immediate return for each time step is defined as follows: if the capital is invested in cash, then there is nothing to earn even if the stock price increases, if the investor has bought stocks the return is equal the relative change of the stock price weighted by the invested amount of capital minus the transaction costs which apply if one changed from cash to stocks.

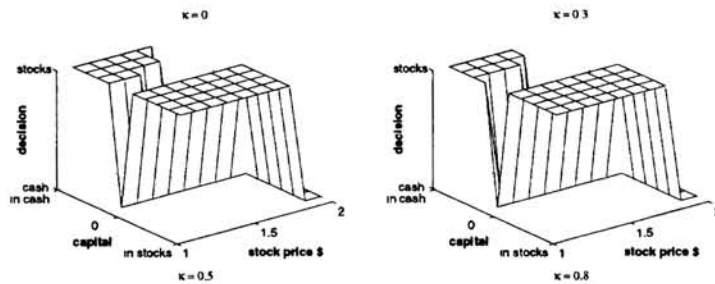

**Figure 3.** Left: Risk neutral policy, $\kappa = 0$. Right: A small bias of $\kappa = 0.3$ against risk changes the policy if one is not invested (transaction costs apply in this case).

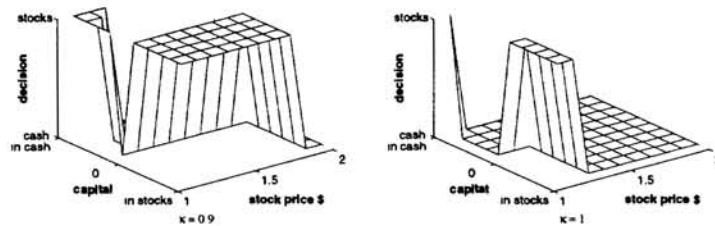

**Figure 4.** Left: $\kappa = 0.5$ yields a stronger risk averse attitude. Right: With $\kappa = 0.8$ the policy becomes also more cautious if already invested in stocks.

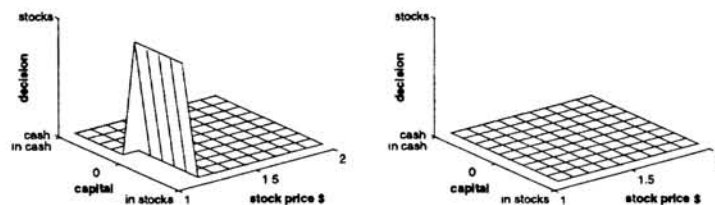

**Figure 5.** Left: $\kappa = 0.9$ leads to a policy which invests in stocks in only 5 cases. Right: The worst case solution never invests because there is always a positive probability for decreasing stock prices.

As a reinforcement learning method, $Q$-learning has to interact with the environment (here the stock market) to learn optimal investment behavior. Thus, a training set of 2000 data points is generated. The training phase is divided into epochs which consists of as many trials as data in the training set exist. At every trial the algorithm selects randomly a stock price from the data set, chooses a random investment state and updates the tabulated $Q$-values according to the procedure given in Neuneier, 1998. The only difference of our new risk averse $Q$-learning is that negative experiences, i.e. smaller returns than in the mean, are overweighted in comparison to positive experiences using the $\kappa$-factor of eq. (7). Using different $\kappa$ values from 0 (recovering the original $Q$-learning procedure) to 1 (leading to worst case $Q$-learning) we plot the resulting policies as mappings from the state space to control actions in figures 3 to 5. Obviously, with increasing $\kappa$ the investor acts more and more cautiously because there are less states associated with an investment decision for stocks. In the extreme case of $\kappa = 1$, there is no stock investment at all in order to avoid any loss. The policy is not useful in practice. This supports our introductory comments that worst case $Q$-learning is not appropriate in many tasks.

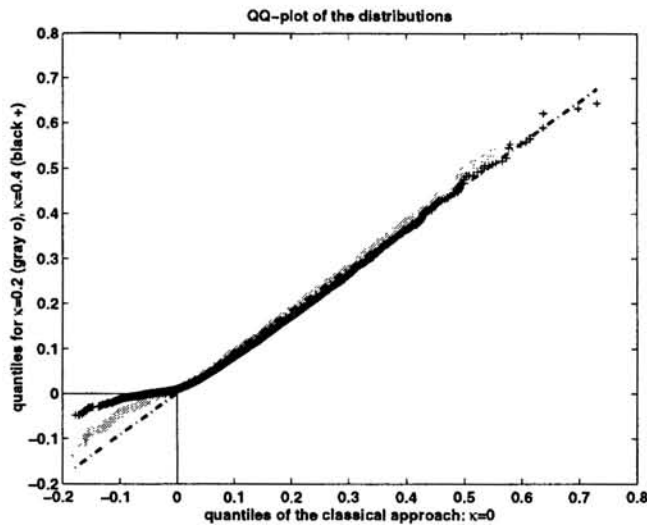

**Figure 6.** The quantiles of the distributions of the discounted sum of returns for $\kappa = 0.2$ (o) and $\kappa = 0.4$ (+) are plotted against the quantiles for the classical risk neutral approach $\kappa = 0$. The distributions only differ significantly for negative accumulated returns (left tail of the distributions).

For further analysis, we specify a risky start state $i_0$ for which a sudden drop of the stock price in the near future is very probable. Starting at $i_0$ we compute the cumulated discounted rewards of 10000 different trajectories following the policies $\pi_0$, $\pi_{0.2}$ and $\pi_{0.4}$ which have been generated using $\kappa = 0$ (risk neutral), $\kappa = 0.2$ and $\kappa = 0.4$. The resulting three data sets are compared using a quantile-quantile plot whose purpose is to determine whether the samples come from the same distribution type. If they do so, the plot will be linear. Fig. 6 clearly shows that for higher $\kappa$-values the left tail of the distribution (negative returns) bends up indicating a fewer number of losses. On the other hand there is no significant difference for positive quantiles. In contrast to naive utility functions which penalizes high variance in general, our risk sensitive $Q$-learning asymmetrically reduces the probability for losses which may be more suitable for many applications.

## 5 CONCLUSION

We have formulated a new $Q$-learning algorithm which can be continuously tuned towards risk seeking or risk avoiding policies. Thus, it is possible to construct control strategies which are more suitable for the problem at hand by only small modifications of $Q$-learning algorithm. The advantage of our approch in comparison to already known solutions is, that we have neither to change the cost nor the state model. We can prove that our algorithm converges under the usual assumptions. Future work will focus on the connections between our approach and the utility theoretic point of view.

**References**

D. P. Bertsekas, J. N. Tsitsiklis (1996) *Neuro-Dynamic Programming.* Athena Scientific.
M. Heger (1994) Consideration of Risk and Reinforcement Learning, in Machine Learning, proceedings of the 11th International Conference, Morgan Kaufmann Publishers.
S. Koenig, R. G. Simmons (1994) Risk-Sensitive Planning with Probabilistic Decision Graphs. Proc. of the Fourth Int. Conf. on Principles of Knowledge Representation and Reasoning (KR).
M. L. Littman, Cs. Szepesvári (1996), A generalized reinforcement-learning model: Convergence and applications. In International Conference of Machine Learning '96. Bari.
R. Neuneier (1998) Enhancing Q-learning for Optimal Asset Allocation, in *Advances in Neural Information Processing Systems 10*, Cambridge, MA: MIT Press.
M. L. Puterman (1994), *Markov Decision Processes*, John Wiley & Sons.
Cs. Szepesvári (1997) Non-Markovian Policies in Sequential Decision Problems, Acta Cybernetica.
J. N. Tsitsiklis, B. Van Roy (1997) Approximate Solutions to Optimal Stopping Problems, in *Advances in Neural Information Processing Systems 9*, Cambridge, MA: MIT Press.